# Neural Network Ensembles, Cross Validation, and Active Learning

**Anders Krogh**\*
Nordita
Blegdamsvej 17
2100 Copenhagen, Denmark

**Jesper Vedelsby**
Electronics Institute, Building 349
Technical University of Denmark
2800 Lyngby, Denmark

## Abstract

Learning of continuous valued functions using neural network ensembles (committees) can give improved accuracy, reliable estimation of the generalization error, and active learning. The *ambiguity* is defined as the variation of the output of ensemble members averaged over *unlabeled* data, so it quantifies the disagreement among the networks. It is discussed how to use the ambiguity in combination with cross-validation to give a reliable estimate of the ensemble generalization error, and how this type of ensemble cross-validation can sometimes improve performance. It is shown how to estimate the optimal weights of the ensemble members using *unlabeled data*. By a generalization of query by committee, it is finally shown how the ambiguity can be used to select new training data to be labeled in an active learning scheme.

## 1 INTRODUCTION

It is well known that a combination of many different predictors can improve predictions. In the neural networks community "ensembles" of neural networks has been investigated by several authors, see for instance [1, 2, 3]. Most often the networks in the ensemble are trained individually and then their predictions are combined. This combination is usually done by majority (in classification) or by simple averaging (in regression), but one can also use a weighted combination of the networks.

At the workshop after the last NIPS conference (December, 1993) an entire session was devoted to ensembles of neural networks ("Putting it all together", chaired by Michael Perrone). Many interesting papers were given, and it showed that this area is getting a lot of attention.

A combination of the output of several networks (or other predictors) is only useful if they disagree on some inputs. Clearly, there is no more information to be gained from a million identical networks than there is from just one of them (see also [2]). By quantifying the disagreement in the ensemble it turns out to be possible to state this insight rigorously for an ensemble used for approximation of real-valued functions (regression). The simple and beautiful expression that relates the disagreement (called the ensemble ambiguity) and the generalization error is the basis for this paper, so we will derive it with no further delay.

## 2   THE BIAS-VARIANCE TRADEOFF

Assume the task is to learn a function $f$ from $R^N$ to $R$ for which you have a sample of $p$ examples, $(x^\mu, y^\mu)$, where $y^\mu = f(x^\mu)$ and $\mu = 1, \ldots, p$. These examples are assumed to be drawn randomly from the distribution $p(x)$. Anything in the following is easy to generalize to several output variables.

The ensemble consists of $N$ networks and the output of network $\alpha$ on input $x$ is called $V^\alpha(x)$. A weighted ensemble average is denoted by a bar, like

$$\overline{V}(x) = \sum_\alpha w_\alpha V^\alpha(x). \tag{1}$$

This is the final output of the ensemble. We think of the weight $w_\alpha$ as our belief in network $\alpha$ and therefore constrain the weights to be positive and sum to one. The constraint on the sum is crucial for some of the following results.

The *ambiguity* on input $x$ of a single member of the ensemble is defined as $a^\alpha(x) = (V^\alpha(x) - \overline{V}(x))^2$. The *ensemble ambiguity* on input $x$ is

$$\overline{a}(x) = \sum_\alpha w_\alpha a^\alpha(x) = \sum_\alpha w_\alpha (V^\alpha(x) - \overline{V}(x))^2. \tag{2}$$

It is simply the variance of the weighted ensemble around the weighed mean, and it measures the disagreement among the networks on input $x$. The quadratic error of network $\alpha$ and of the ensemble are

$$\epsilon^\alpha(x) = (f(x) - V^\alpha(x))^2 \tag{3}$$

$$e(x) = (f(x) - \overline{V}(x))^2 \tag{4}$$

respectively. Adding and subtracting $f(x)$ in (2) yields

$$\overline{a}(x) = \sum_\alpha w_\alpha \epsilon^\alpha(x) - e(x) \tag{5}$$

(after a little algebra using that the weights sum to one). Calling the weighted average of the individual errors $\overline{\epsilon}(x) = \sum_\alpha w_\alpha \epsilon^\alpha(x)$ this becomes

$$e(x) = \overline{\epsilon}(x) - \overline{a}(x). \tag{6}$$

All these formulas can be averaged over the input distribution. Averages over the input distribution will be denoted by capital letter, so

$$E^\alpha = \int \mathrm{d}x p(x) \epsilon^\alpha(x) \tag{7}$$

$$A^\alpha = \int \mathrm{d}x p(x) a^\alpha(x) \tag{8}$$

$$E = \int \mathrm{d}x p(x) e(x). \tag{9}$$

The first two of these are the generalization error and the ambiguity respectively for network $\alpha$, and $E$ is the generalization error for the ensemble. From (6) we then find for the *ensemble generalization error*

$$E = \overline{E} - \overline{A}. \tag{10}$$

The first term on the right is the weighted average of the generalization errors of the individual networks $(\overline{E} = \sum_\alpha w_\alpha E^\alpha)$, and the second is the weighted average of the ambiguities $(\overline{A} = \sum_\alpha w_\alpha A^\alpha)$, which we refer to as the ensemble ambiguity.

The beauty of this equation is that it separates the generalization error into a term that depends on the generalization errors of the individual networks and another term that contain *all correlations* between the networks. Furthermore, the correlation term $\overline{A}$ can be estimated entirely from *unlabeled data, i.e.*, no knowledge is required of the real function to be approximated. The term "unlabeled example" is borrowed from classification problems, and in this context it means an input $x$ for which the value of the target function $f(x)$ is unknown.

Equation (10) expresses the tradeoff between bias and variance in the ensemble, but in a different way than the the common bias-variance relation [4] in which the averages are over possible training sets instead of ensemble averages. If the ensemble is strongly biased the ambiguity will be small, because the networks implement very similar functions and thus agree on inputs even outside the training set. Therefore the generalization error will be essentially equal to the weighted average of the generalization errors of the individual networks. If, on the other hand, there is a large variance, the ambiguity is high and in this case the generalization error will be smaller than the average generalization error. See also [5].

From this equation one can immediately see that the generalization error of the ensemble is always smaller than the (weighted) average of the ensemble errors, $E < \overline{E}$. In particular for uniform weights:

$$E \leq \frac{1}{N} \sum_\alpha E^\alpha \tag{11}$$

which has been noted by several authors, see *e.g.* [3].

## 3   THE CROSS-VALIDATION ENSEMBLE

From (10) it is obvious that increasing the ambiguity (while not increasing individual generalization errors) will improve the overall generalization. We want the networks to disagree! How can we increase the ambiguity of the ensemble? One way is to use different types of approximators like a mixture of neural networks of different topologies or a mixture of completely different types of approximators. Another

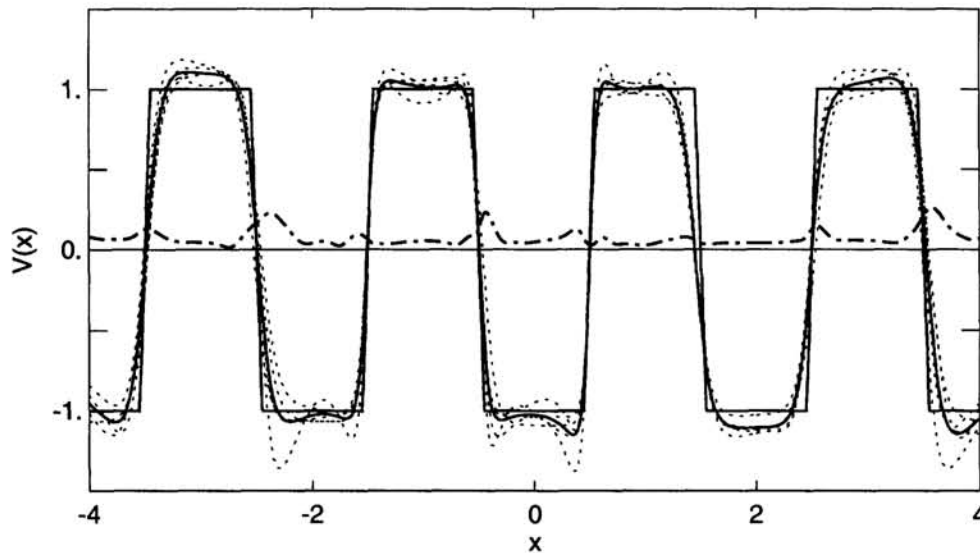

Figure 1: An ensemble of five networks were trained to approximate the square wave target function $f(x)$. The final ensemble output (solid smooth curve) and the outputs of the individual networks (dotted curves) are shown. Also the square root of the ambiguity is shown (dash-dot line). For training 200 random examples were used, but each network had a cross-validation set of size 40, so they were each trained on 160 examples.

obvious way is to train the networks on different training sets. Furthermore, to be able to estimate the first term in (10) it would be desirable to have some kind of cross-validation. This suggests the following strategy.

Chose a number $K \leq p$. For each network in the ensemble hold out $K$ examples for testing, where the $N$ test sets should have minimal overlap, *i.e.*, the $N$ training sets should be as different as possible. If, for instance, $K \leq p/N$ it is possible to choose the $K$ test sets with no overlap. This enables us to estimate the generalization error $E^\alpha$ of the individual members of the ensemble, and at the same time make sure that the ambiguity increases. When holding out examples the generalization errors for the individual members of the ensemble, $E^\alpha$, will increase, but the conjecture is that for a good choice of the size of the ensemble ($N$) and the test set size ($K$), the ambiguity will increase more and thus one will get a decrease in overall generalization error.

This conjecture has been tested experimentally on a simple square wave function of one variable shown in Figure 1. Five identical feed-forward networks with one hidden layer of 20 units were trained independently by back-propagation using 200 random examples. For each network a cross-validation set of $K$ examples was held out for testing as described above. The "true" generalization and the ambiguity were estimated from a set of 1000 random inputs. The weights were uniform, $w^\alpha = 1/5$ (non-uniform weights are addressed later).

In Figure 2 average results over 12 independent runs are shown for some values of

Figure 2: The solid line shows the generalization error for uniform weights as a function of $K$, where $K$ is the size of the cross-validation sets. The dotted line is the error estimated from equation (10). The dashed line is for the optimal weights estimated by the use of the generalization errors for the individual networks estimated from the cross-validation sets as described in the text. The bottom solid line is the generalization error one would obtain if the individual generalization errors were known exactly (the best possible weights).

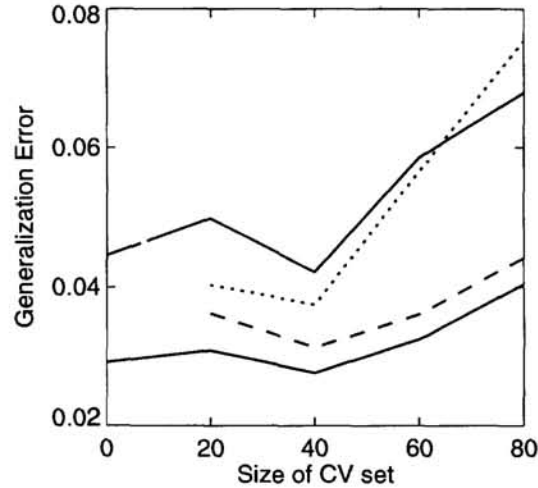

$K$ (top solid line). First, one should note that the generalization error is the same for a cross-validation set of size 40 as for size 0, although not lower, so it supports the conjecture in a weaker form. However, we have done many experiments, and depending on the experimental setup the curve can take on almost any form, sometimes the error is larger at zero than at 40 or vice versa. In the experiments shown, only ensembles with at least four converging networks out of five were used. If all the ensembles were kept, the error would have been significantly higher at $K = 0$ than for $K > 0$ because in about half of the runs none of the networks in the ensemble converged — something that seldom happened when a cross-validation set was used. Thus it is still unclear under which circumstances one can expect a drop in generalization error when using cross-validation in this fashion.

The dotted line in Figure 2 is the error estimated from equation (10) using the cross-validation sets for each of the networks to estimate $E^\alpha$, and one notices a good agreement.

## 4 OPTIMAL WEIGHTS

The weights $w_\alpha$ can be estimated as described in *e.g.* [3]. We suggest instead to use unlabeled data and estimate them in such a way that they minimize the generalization error given in (10).

There is no analytical solution for the weights, but something can be said about the minimum point of the generalization error. Calculating the derivative of $E$ as given in (10) subject to the constraints on the weights and setting it equal to zero shows that

$$E^\alpha - A^\alpha = E \text{ or } w_\alpha = 0. \tag{12}$$

(The calculation is not shown because of space limitations, but it is easy to do.) That is, $E^\alpha - A^\alpha$ has to be the same for all the networks. Notice that $A^\alpha$ depends on the weights through the ensemble average of the outputs. It shows that the optimal weights have to be chosen such that each network contributes exactly $w_\alpha E$

to the generalization error. Note, however, that a member of the ensemble can have such a poor generalization or be so correlated with the rest of the ensemble that it is optimal to set its weight to zero.

The weights can be "learned" from *unlabeled examples, e.g.* by gradient descent minimization of the estimate of the generalization error (10). A more efficient approach to finding the optimal weights is to turn it into a quadratic optimization problem. That problem is non-trivial only because of the constraints on the weights ($\sum_\alpha w_\alpha = 1$ and $w_\alpha \geq 0$). Define the correlation matrix,

$$C_{\alpha\beta} = \int \mathrm{d}x p(x) V^\alpha(x) V^\beta(x). \tag{13}$$

Then, using that the weights sum to one, equation (10) can be rewritten as

$$E = \sum_\alpha w_\alpha E^\alpha + \sum_{\alpha\beta} w_\alpha C_{\alpha\beta} w_\beta - \sum_\alpha w_\alpha C_{\alpha\alpha}. \tag{14}$$

Having estimates of $E^\alpha$ and $C_{\alpha\beta}$ the optimal weights can be found by linear programming or other optimization techniques. Just like the ambiguity, the correlation matrix can be estimated from *unlabeled data* to any accuracy needed (provided that the input distribution $p$ is known).

In Figure 2 the results from an experiment with weight optimization are shown. The dashed curve shows the generalization error when the weights are optimized as described above using the estimates of $E^\alpha$ from the cross-validation (on K examples). The lowest solid curve is for the idealized case, when it is assumed that the errors $E^\alpha$ are known exactly, so it shows the lowest possible error. The performance improvement is quite convincing when the cross-validation estimates are used.

It is important to notice that any estimate of the generalization error of the individual networks can be used in equation (14). If one is certain that the individual networks do not overfit, one might even use the training errors as estimates for $E^\alpha$ (see [3]). It is also possible to use some kind of regularization in (14), if the cross-validation sets are small.

## 5   ACTIVE LEARNING

In some neural network applications it is very time consuming and/or expensive to acquire training data, *e.g.*, if a complicated measurement is required to find the value of the target function for a certain input. Therefore it is desirable to only use examples with maximal information about the function. Methods where the learner points out good examples are often called active learning.

We propose a query-based active learning scheme that applies to ensembles of networks with continuous-valued output. It is essentially a generalization of query by committee [6, 7] that was developed for classification problems. Our basic assumption is that those patterns in the input space yielding the largest error are those points we would benefit the most from including in the training set.

Since the generalization error is always non-negative, we see from (6) that the weighted average of the individual network errors is always larger than or equal to the ensemble ambiguity,

$$\bar{\epsilon}(x) \geq \bar{a}(x), \tag{15}$$

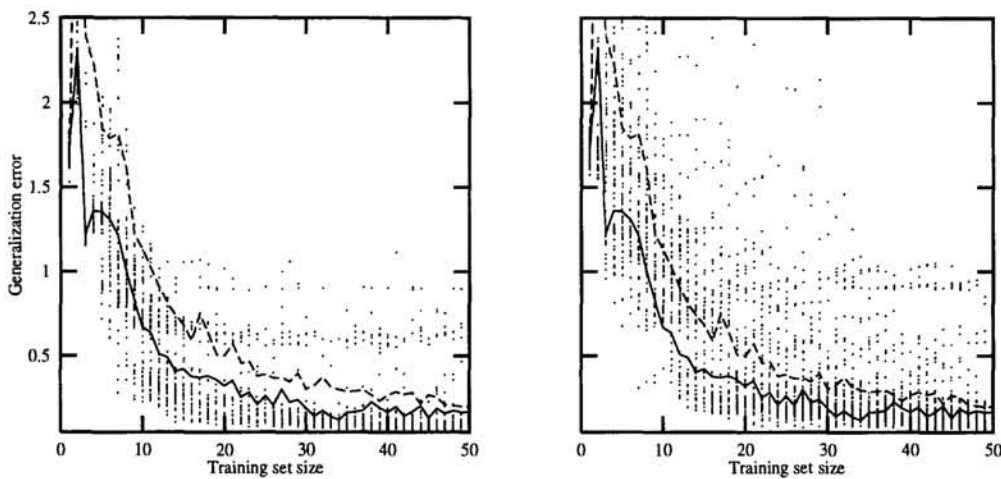

Figure 3: In both plots the full line shows the average generalization for *active learning*, and the dashed line for *passive learning* as a function of the number of training examples. The dots in the left plot show the results of the individual experiments contributing to the mean for the active learning. The dots in right plot show the same for passive learning.

which tells us that the ambiguity is a lower bound for the weighted average of the squared error. An input pattern that yields a large ambiguity will always have a large average error. On the other hand, a low ambiguity does not necessarily imply a low error. If the individual networks are trained to a low training error on the same set of examples then both the error and the ambiguity are low on the training points. This ensures that a pattern yielding a large ambiguity cannot be in the close neighborhood of a training example. The ambiguity will to some extent follow the fluctuations in the error. Since the ambiguity is calculated from unlabeled examples the input-space can be scanned for these areas to any detail. These ideas are well illustrated in Figure 1, where the correlation between error and ambiguity is quite strong, although not perfect.

The results of an experiment with the active learning scheme is shown in Figure 3. An ensemble of 5 networks was trained to approximate the square-wave function shown in Figure 1, but in this experiments the function was restricted to the interval from −2 to 2. The curves show the final generalization error of the ensemble in a passive (dashed line) and an active learning test (solid line). For each training set size 2x40 independent tests were made, all starting with the same initial training set of a single example. Examples were generated and added one at a time. In the passive test examples were generated at random, and in the active one each example was selected as the input that gave the largest ambiguity out of 800 random ones. Figure 3 also shows the distribution of the individual results of the active and passive learning tests. Not only do we obtain significantly better generalization by active learning, there is also less scatter in the results. It seems to be easier for the ensemble to learn from the actively generated set.

# 6   CONCLUSION

The central idea in this paper was to show that there is a lot to be gained from using unlabeled data when training in ensembles. Although we dealt with neural networks, all the theory holds for any other type of method used as the individual members of the ensemble.

It was shown that apart from getting the individual members of the ensemble to generalize well, it is important for generalization that the individuals disagrees as much as possible, and we discussed one method to make even identical networks disagree. This was done by training the individuals on different training sets by holding out some examples for each individual during training. This had the added advantage that these examples could be used for testing, and thereby one could obtain good estimates of the generalization error.

It was discussed how to find the optimal weights for the individuals of the ensemble. For our simple test problem the weights found improved the performance of the ensemble significantly.

Finally a method for active learning was described, which was based on the method of query by committee developed for classification problems. The idea is that if the ensemble disagrees strongly on an input, it would be good to find the label for that input and include it in the training set for the ensemble. It was shown how active learning improves the learning curve a lot for a simple test problem.

### Acknowledgements

We would like to thank Peter Salamon for numerous discussions and for his implementation of linear programming for optimization of the weights. We also thank Lars Kai Hansen for many discussions and great insights, and David Wolpert for valuable comments.

## Footnotes

\*Author to whom correspondence should be addressed. Email: krogh@nordita.dk

# References

[1] L.K. Hansen and P Salamon. Neural network ensembles. *IEEE Transactions on Pattern Analysis and Machine Intelligence*, 12(10):993–1001, Oct. 1990.

[2] D.H Wolpert. Stacked generalization. *Neural Networks*, 5(2):241–59, 1992.

[3] Michael P. Perrone and Leon N Cooper. When networks disagree: Ensemble method for neural networks. In R. J. Mammone, editor, *Neural Networks for Speech and Image processing*. Chapman-Hall, 1993.

[4] S. Geman, E. Bienenstock, and R Doursat. Neural networks and the bias/variance dilemma. *Neural Computation*, 4(1):1–58, Jan. 1992.

[5] Ronny Meir. Bias, variance and the combination of estimators; the case of linear least squares. Preprint (In Neuroprose), Technion, Heifa, Israel, 1994.

[6] H.S. Seung, M. Opper, and H. Sompolinsky. Query by committee. In *Proceedings of the Fifth Workshop on Computational Learning Theory*, pages 287–294, San Mateo, CA, 1992. Morgan Kaufmann.

[7] Y. Freund, H.S. Seung, E. Shamir, and N. Tishby. Information, prediction, and query by committee. In *Advances in Neural Information Processing Systems*, volume 5, San Mateo, California, 1993. Morgan Kaufmann.